# VLSI Implementation of a High-Capacity Neural Network Associative Memory

Tzi-Dar Chiueh [1] and Rodney M. Goodman
Department of Electrical Engineering (116-81)
California Institute of Technology
Pasadena, CA 91125, USA

## ABSTRACT

In this paper we describe the VLSI design and testing of a high capacity associative memory which we call the exponential correlation associative memory (ECAM). The prototype $3\mu$-CMOS programmable chip is capable of storing 32 memory patterns of 24 bits each. The high capacity of the ECAM is partly due to the use of special exponentiation neurons, which are implemented via sub-threshold MOS transistors in this design. The prototype chip is capable of performing one associative recall in 3 $\mu$s.

## 1  ARCHITECTURE

Previously (Chiueh, 1989), we have proposed a general model for correlation-based associative memories, which includes a variant of the Hopfield memory and high-order correlation memories as special cases. This new exponential correlation associative memory (ECAM) possesses a very large storage capacity, which scales *exponentially* with the length of memory patterns (Chiueh, 1988). Furthermore, it has been shown that the ECAM is asymptotically stable in both synchronous and

asynchronous updating modes (Chiueh, 1989). The model is based on an architecture consisting of binary connection weights, simple hard-limiter neurons, and specialized nonlinear circuits as shown in Figure 1. The evolution equation of this general model is

$$\mathbf{x}' = sgn\left\{ \sum_{k=1}^{M} f(<\mathbf{u}^{(k)}, \mathbf{x}>)\, \mathbf{u}^{(k)} \right\}, \tag{1}$$

where $\mathbf{u}^{(1)}, \mathbf{u}^{(2)}, \cdots, \mathbf{u}^{(M)}$ are the $M$ memory patterns. $\mathbf{x}$ and $\mathbf{x}'$ are the current and the next state patterns of the system respectively, and $sgn$ is the threshold function, which takes on the value $+1$ if its argument is nonnegative, and $-1$ otherwise.

We addressed, in particular, the case where $f(\cdot)$ is in the form of an exponentiation, namely, when the evolution equation is given by

$$\mathbf{x}' = sgn\left\{ \sum_{k=1}^{M} a^{<\mathbf{u}^{(k)}, \mathbf{x}>}\, \mathbf{u}^{(k)} \right\}, \tag{2}$$

and $a$ is a constant greater than unity.

The ECAM chip we have designed is *programmable*; that is, one can change the stored memory patterns at will. To perform an associative recall, one first loads a set of memory patterns into the chip. The chip is then switched to the associative recall mode, an input pattern is presented to the ECAM chip, and the ECAM chip then computes the next state pattern according to Equation (2). The components of the next state pattern appear at the output in parallel after the internal circuits have settled. Feedback is easily incorporated by connecting the output port to the input port, in which case the chip will cycle until a fixed point is reached.

## 2   DESIGN OF THE ECAM CIRCUITS

From the evolution equation of the ECAM, we notice that there are essentially three circuits that need to be designed in order to build an ECAM chip. They are:

- $<\mathbf{u}^{(k)}, \mathbf{x}>$, the correlation computation circuit;
- $\sum_{k=1}^{M} a^{<\mathbf{u}^{(k)}, \mathbf{x}>}\, \mathbf{u}^{(k)}$, the exponentiation, multiplication and summing circuit;
- $sgn(\cdot)$, the threshold circuit.

We now describe each circuit, present its design, and finally integrate all these circuits to get the complete design of the ECAM chip.

## 2.1   CORRELATION COMPUTATION

In Figure 2, we illustrate a voltage-divider type circuit consisting of NMOS transistors working as controlled resistors (linear resistors or open circuits). This circuit computes the correlation between the input pattern $\mathbf{x}$ and a memory pattern $\mathbf{u}^{(k)}$. If the $i^{\text{th}}$ components of these two patterns are the same, the corresponding XOR gate outputs a "0" and there is a connection from the node $V_{ux}^{(k)}$ to VBB; otherwise, there is a connection from $V_{ux}^{(k)}$ to GND. Hence the output voltage will be proportional to the number of positions at which $\mathbf{x}$ and $\mathbf{u}^{(k)}$ match. The maximum output voltage is controlled by an externally supplied bias voltage VBB. Normally, VBB is set to a voltage lower than the threshold voltage of NMOS transistors (VTH) for a reason that will be explained later. Note that the conductance of an NMOS transistor in the ON mode is not fixed, but rather depends on its gate-to-source voltage and its drain-to-source voltage. Thus, some nonlinearity is bound to occur in the correlation computation circuit, however, simulation shows that this effect is small.

## 2.2   EXPONENTIATION, MULTIPLICATION, AND SUMMATION

Figure 4 shows a circuit that computes the exponentiation of $V_{ux}^{(k)}$, the product of the $u_i^{(k)}$ and the exponential, and the sum of all M products.

The exponentiation function is implemented by an NMOS transistor whose gate voltage is $V_{ux}^{(k)}$. Since VBB, the maximum value that $V_{ux}^{(k)}$ can assume, is set to be lower than the threshold voltage (VTH); the NMOS transistor is in the subthreshold region, where its drain current depends exponentially on its gate-to-source voltage (Mead, 1989). If we temporarily ignore the transistors controlled by $u_i^{(k)}$ or the complement of $u_i^{(k)}$, the current flowing through the exponentiation transistor associated with $V_{ux}^{(k)}$ will scale exponentially with $V_{ux}^{(k)}$. Therefore, the exponentiation function is properly computed.

Since the multiplier $u_i^{(k)}$ assumes either $+1$ or $-1$, the multiplication can be easily done by forming two branches, each made up of a transmission gate in series with an exponentiation transistor whose gate voltage is $V_{ux}^{(k)}$. One of the two transmission gates is controlled by $u_i^{(k)}$, and the other by the complement of $u_i^{(k)}$. Consequently, when $u_i^{(k)} = 1$, the positive branch will carry a current that scales exponentially with the correlation of the input $\mathbf{x}$ and the $k^{\text{th}}$ memory pattern $\mathbf{u}^{(k)}$, while the negative branch is essentially an open circuit, and vice versa.

Summation of the $M$ terms in the evolution equation is done by current summing. The final results are two currents $I_i^+$ and $I_i^-$, which need to be compared by a threshold circuit to determine the sign of the $i^{\text{th}}$ bit of the next state pattern $x_i'$. In the ECAM a simple differential amplifier (Figure 3) performs the comparison.

## 2.3   THE BASIC ECAM CELL

The above computational circuits are then combined with a simple static RAM cell, to make up a basic ECAM cell as illustrated in Figure 5. The final design of an ECAM that stores $M$ $N$-bit memory patterns can be obtained by replicating the basic ECAM cell $M$ times in the horizontal direction and $N$ times in the vertical direction, together with read/write circuits, sense amplifiers, address decoders, and I/O multiplexers. The prototype ECAM chip is made up of 32 × 24 ECAM cells, and stores 32 memory patterns each 24 bits wide.

## 3   ECAM CHIP TEST RESULTS

The test procedure for the ECAM is to first generate 32 memory patterns at random and then program the ECAM chip with these 32 patterns. We then pick a memory pattern at random, flip a specified number of bits randomly, and feed the resulting pattern to the ECAM as an input pattern ($\mathbf{x}$). The output pattern ($\mathbf{x}'$) can then be fed back to the inputs of the ECAM chip. This iteration continues until the pattern at the input is the same as that at the the output, at which time the ECAM chip is said to have reached a stable state. We select 10 sets of 32 memory patterns and for each set we run the ECAM chip on 100 trial input patterns with a fixed number of errors. Altogether, the test consists of 1000 trials.

In Figure 6, we illustrate the ECAM chip test results. The number of successes is plotted against the number of errors in the input patterns for the following four cases: 1) The ECAM chip with $V_{BB} = 5V$; 2) $V_{BB} = 2V$; 3) $V_{BB} = 1V$; and 4) a simulated ECAM in which the exponentiation constant $a$, equals 2. It is apparent from Figure 6 that as the number of errors increases, the number of successes decreases, which is expected. Also, one notices that the simulated ECAM is by far the best one, which is again not unforeseen because the ECAM chip is, after all, only an approximation of the ideal ECAM model.

What is really unexpected is that the best performance occurs for $V_{BB} = 2V$ rather than $V_{BB} = 1V$ ($V_{TH}$ in this CMOS process). This phenomenon arises because of two contradictory effects brought about by increasing $V_{BB}$. On the one hand, increasing $V_{BB}$ increases the dynamic range of the exponentiation transistors in the ECAM chip. Suppose that the correlations of two memory patterns $\mathbf{u}^{(l)}$ and $\mathbf{u}^{(k)}$ with the input pattern $\mathbf{x}$ are $t_l$ and $t_k$, respectively, where $t_l > t_k$; then

$$V_{ux}^{(l)} = \frac{(t_l + N)\, V_{BB}}{2N}, \quad V_{ux}^{(k)} = \frac{(t_k + N)\, V_{BB}}{2N}.$$

Therefore, as $V_{BB}$ increases, so does the difference between $V_{ux}^{(l)}$ and $V_{ux}^{(k)}$, and $\mathbf{u}^{(l)}$ becomes more dominant than $\mathbf{u}^{(k)}$ in the weighted sum of the evolution equation.

Hence, as VBB increases, the error correcting ability of the ECAM chip should improve. On the other hand, as VBB increases beyond the threshold voltage, the exponentiation transistors leave the subthreshold region and may enter saturation, where the drain current is approximately proportional to the *square* of the gate-to-source voltage. Since a second-order correlation associative memory in general possesses a smaller storage capacity than an ECAM, one would expect that with a fixed number of loaded memory patterns, the ECAM should do better than the second-order correlation associative memory. Thus one effect tends to enhance the performance of the ECAM chip, while the other tends to degrade it. A compromise between these two effects is reached, and the best performance is achieved when VBB = 2V.

For the case when VBB = 2V, the drain current versus gate-to-source voltage characteristic of the exponentiation transistors is actually a hybrid of a square function and an exponentiation function. At the bottom it is of an exponential form, and it gradually flattens out to a square function, once the gate-to-source voltage becomes larger than the threshold voltage. Therefore, the ECAM chip with VBB = 2V is a mixture of the second-order correlation associative memory and the pure ECAM. According to the convergence theorem for correlation associative memories (Chiueh, 1989) and the fact that $f(\cdot)$ in the ECAM chip with VBB = 2V is still monotonically nondecreasing, the ECAM chip is still asymptotically stable when VBB = 2V.

We have tested the speed of the ECAM chip using binary image vector quantization as an example problem. The speed at which the ECAM chip can vector-quantize binary images is of interest. We find experimentally that the ECAM chip is capable of doing one associative recall operation, in less than 3 $\mu$s, on 4 × 4 blocks. This projects to approximately 49 ms for a 512 × 512 binary image, or more than 20 images per second.

## 4   CONCLUSIONS

In this paper, we have presented a VLSI circuit design for implementing a high capacity correlation associative memory. The performance of the ECAM chip is shown to be almost as good as a computer-simulated ECAM. Furthermore, we believe that the ECAM chip is more robust than an associative memory using a winner-take-all function, because it obtains its result via iteration, as opposed to one shot. In conclusion, we believe that the ECAM chip provides a fast and efficient way for solving many associative recall problems, such as vector quantization and optical character recognition.

**Acknowledgement**

This work was supported in part by NSF grant No. MIP − 8711568.

## Footnotes

[1] Tzi-Dar Chiueh is now with the Department of Electrical Engineering, National Taiwan University, Taipei, Taiwan 10764.

## References

T. D. Chiueh and R. M. Goodman. (1988) "High Capacity Exponential Associative Memory," in *Proc. of IEEE ICNN*, Vol. I, pp. 153–160.

T. D. Chiueh. (1989) "Pattern Classification and Associative Recall by Neural Networks," Ph. D. dissertation, California Institute of Technology.

C. A. Mead. (1989) *Analog VLSI and Neural Systems.* Reading, MA : Addison-Wesley.

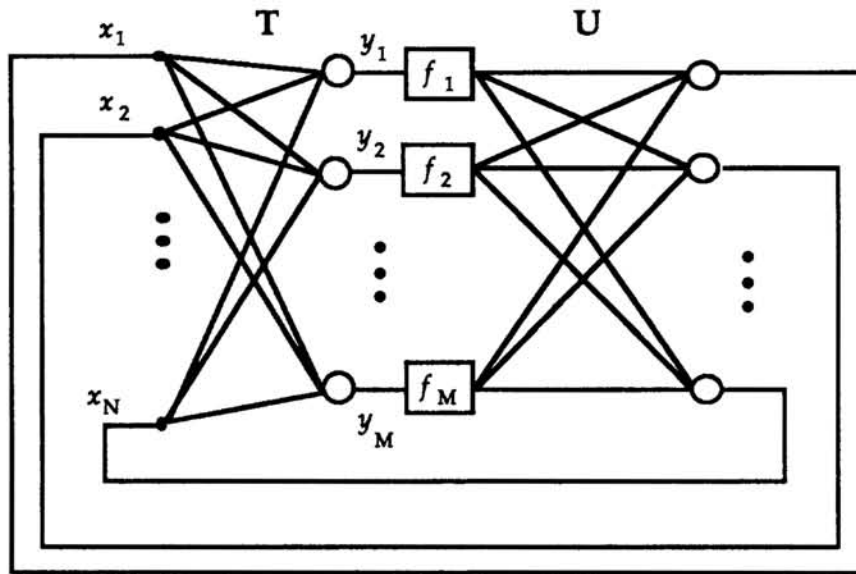

Figure 1: Architecture of the General Correlation-Based Associative Memory

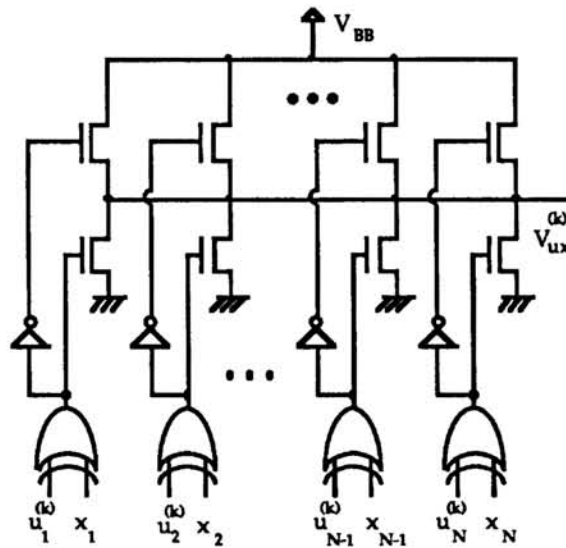

Figure 2: The Correlation Computation Circuit

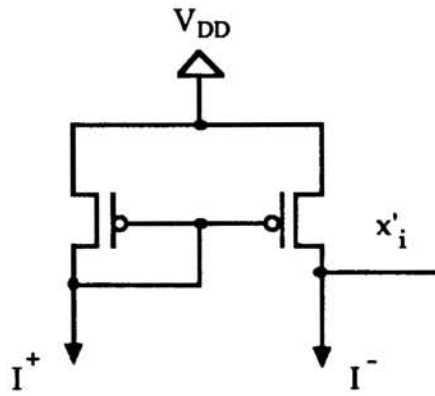

Figure 3: The Threshold Circuit

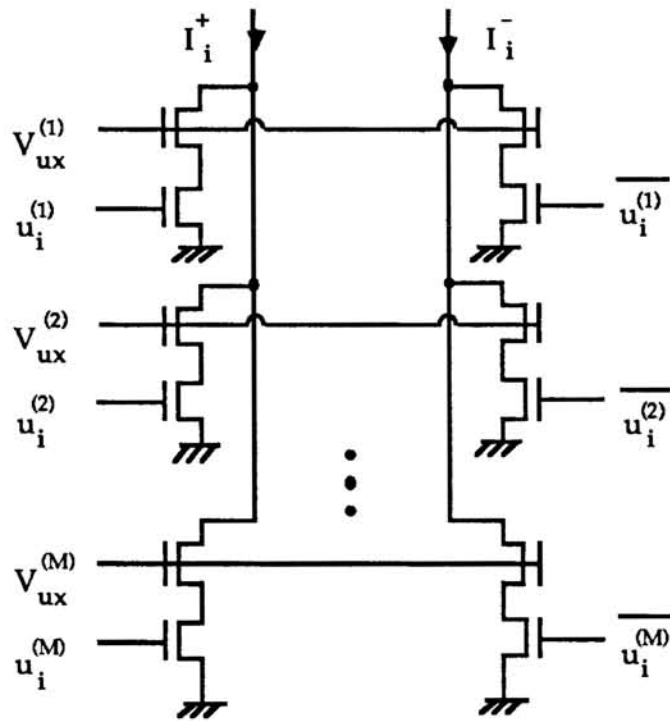

Figure 4: The Exponentiation, Multiplication, and Summation Circuit

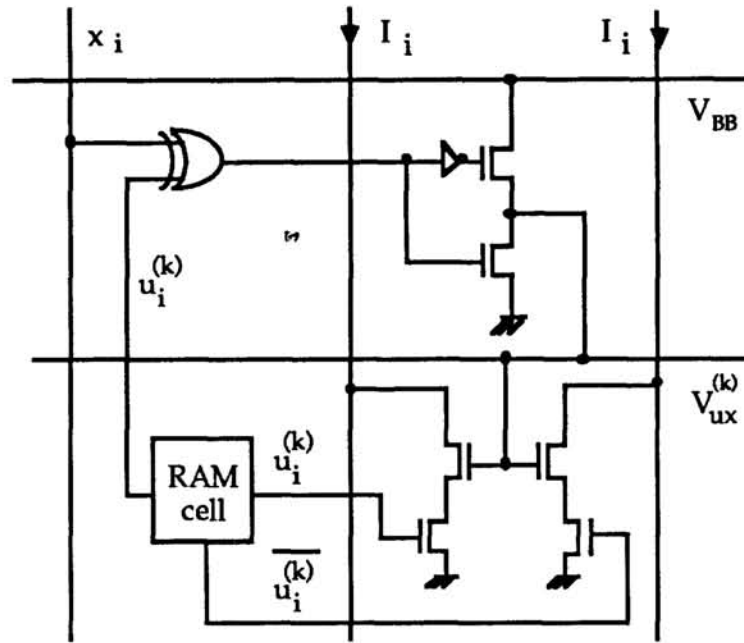

Figure 5: Circuit Diagram of the Basic ECAM Cell

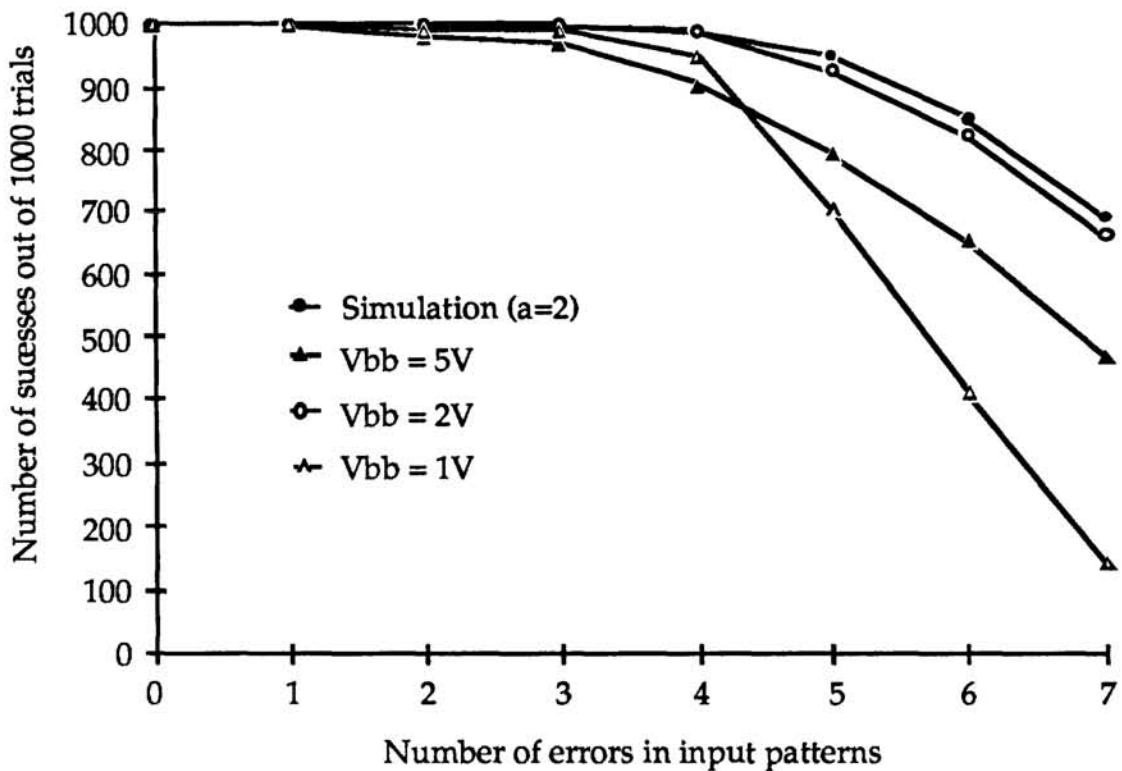

Figure 6: Error Correcting Ability of the ECAM Chip with Different V$_{BB}$ compared with a Simulated ECAM with $a = 2$